# Efficient Kernel Discriminant Analysis via QR Decomposition

**Tao Xiong**
Department of ECE
University of Minnesota
txiong@ece.umn.edu

**Jieping Ye**
Department of CSE
University of Minnesota
jieping@cs.umn.edu

**Qi Li**
Department of CIS
University of Delaware
qili@cis.udel.edu

**Vladimir Cherkassky**
Department of ECE
University of Minnesota
cherkass@ece.umn.edu

**Ravi Janardan**
Department of CSE
University of Minnesota
janardan@cs.umn.edu

## Abstract

Linear Discriminant Analysis (LDA) is a well-known method for feature extraction and dimension reduction. It has been used widely in many applications such as face recognition. Recently, a novel LDA algorithm based on QR Decomposition, namely LDA/QR, has been proposed, which is competitive in terms of classification accuracy with other LDA algorithms, but it has much lower costs in time and space. However, LDA/QR is based on linear projection, which may not be suitable for data with nonlinear structure. This paper first proposes an algorithm called KDA/QR, which extends the LDA/QR algorithm to deal with nonlinear data by using the kernel operator. Then an efficient approximation of KDA/QR called AKDA/QR is proposed. Experiments on face image data show that the classification accuracy of both KDA/QR and AKDA/QR are competitive with Generalized Discriminant Analysis (GDA), a general kernel discriminant analysis algorithm, while AKDA/QR has much lower time and space costs.

## 1 Introduction

Linear Discriminant Analysis [3] is a well–known method for dimension reduction. It has been used widely in many applications such as face recognition [2]. Classical LDA aims to find optimal transformation by minimizing the within-class distance and maximizing the between-class distance simultaneously, thus achieving maximum discrimination. The optimal transformation can be readily computed by computing the eigen-decomposition on the scatter matrices.

Although LDA works well for linear problems, it may be less effective when severe non-linearity is involved. To deal with such a limitation, nonlinear extensions through kernel functions have been proposed. The main idea of kernel-based methods is to map the input data to a feature space through a nonlinear mapping, where the inner products in the feature

space can be computed by a kernel function without knowing the nonlinear mapping explicitly [9]. Kernel Principal Component Analysis (KPCA) [10], Kernel Fisher Discriminant Analysis (KFDA) [7] and Generalized Discriminant Analysis (GDA) [1] are, respectively, kernel-based nonlinear extensions of the well known PCA, FDA and LDA methods.

To our knowledge, there are few efficient algorithms for general kernel based discriminant algorithms — most known algorithms effectively scale as $O(n^3)$ where $n$ is the sample size. In [6, 8], S. Mika *et al.* made a first attempt to speed up KFDA through a greedy approximation technique. However the algorithm was developed to handle the binary classification problem. For multi-class problem, the authors suggested the *one against the rest scheme* by considering all two-class problems.

Recently, an efficient variant of LDA, namely LDA/QR, was proposed in [11, 12]. The essence of LDA/QR is the utilization of QR-decomposition on a small size matrix. The time complexity of LDA/QR is linear in the size of the training data, as well as the number of dimensions of the data. Moreover, experiments in [11, 12] show that the classification accuracy of LDA/QR is competitive with other LDA algorithms.

In this paper, we first propose an algorithm, namely KDA/QR[1], which is a nonlinear extension of LDA/QR. Since KDA/QR involves the whole kernel matrix, which is not scalable for large datasets, we also propose an approximation of KDA/QR, namely AKDA/QR. A distinct property of AKDA/QR is that it scales as $O(ndc)$, where $n$ is the size of the data, $d$ is the dimension of the data, and $c$ is the number of classes.

We apply the proposed algorithms on face image datasets and compare them with LDA/QR, and Generalized Discriminant Analysis (GDA) [1], a general method for kernel discriminant analysis. Experiments show that: (1) AKDA/QR is competitive with KDA/QR and GDA in classification; (2) both KDA/QR and AKDA/QR outperform LDA/QR in classification; and (3) AKDA/QR has much lower costs in time and space than GDA.

## 2   LDA/QR

In this section, we give a brief review of the LDA/QR algorithm [11, 12]. This algorithm has two stages. The first stage maximizes the separation between different classes via QR Decomposition [4]. The second stage addresses the issue of minimizing the within-class distance, while maintaining low time/space complexity.

Let $A \in \mathbb{R}^{d \times n}$ be the data matrix, where each column $a_i$ is a vector in $d$-dimensional space. Assume $A$ is partitioned into $c$ classes $\{\Pi_i\}_{i=1}^{c}$, and the size of the $i$th class $|\Pi_i| = n_i$.

Define *between-class*, *within-class*, and *total scatter* matrices $S_b$, $S_w$, and $S_t$ respectively, as follows [3]: $S_b = H_b H_b^t$, $S_w = H_w H_w^t$, and $S_t = H_t H_t^t$, where $H_b = [\sqrt{N_1}(m_1 - m), \cdots, \sqrt{N_c}(m_c - m)] \in R^{d \times c}$, $H_w = A - [m_1 e_1^t, \cdots, m_c e_c^t] \in R^{d \times n}$, and $H_t = A - me^t \in R^{d \times n}$, $e_i = (1, \cdots, 1)^t \in R^{n_i \times 1}$, $e = (1, \cdots, 1)^t \in R^{n \times 1}$, $m_i$ is the mean of the $i$th class, and $m$ is the global mean. It is easy to check that $S_t = S_b + S_w$.

The first stage of LDA/QR aims to solve the following optimization problem,

$$G = \arg \max_{G^t G = I} \text{trace}(G^t S_b G). \tag{1}$$

Note that this optimization only addresses the issue of maximizing the between-class distance. The solution can be obtained by solving the eigenvalue problem on $S_b$. The solution can also be obtained through QR Decomposition on the centroid matrix $C$ [12], where $C = [m_1, m_2, \cdots, m_c]$ consists of the $c$ centroids. More specifically, let $C = QR$ be the QR Decomposition of $C$, where $Q \in \mathbb{R}^{n \times c}$ has orthonormal columns and $R \in \mathbb{R}^{c \times c}$

**Algorithm 1: LDA/QR**

/* Stage I: */
1. Construct centroid matrix $C$;
2. Compute QR Decomposition of $C$ as $C = QR$, where $Q \in \mathbb{R}^{d \times c}$, $R \in \mathbb{R}^{c \times c}$;
/* Stage II: */
3. $Y \leftarrow H_b^t Q$;
4. $Z \leftarrow H_t^t Q$;
5. $B \leftarrow Y^t Y$; /*Reduced between-class scatter matrix*/
6. $T \leftarrow Z^t Z$; /*Reduced total scatter matrix*/
7. Compute the $c$ eigenvectors $\phi_i$ of $(T + \mu I_c)^{-1} B$ with decreasing eigenvalues;
8. $G \leftarrow QV$, where $V = [\phi_1, \cdots, \phi_c]$.

---

is upper triangular. Then $G = QV$, for any orthogonal matrix $V$, solves the optimization problem in Eq. (1). Note that the choice of orthogonal matrix $V$ is arbitrary, since $\text{trace}(G^t S_b G) = \text{trace}(V^t G^t S_b GV)$, for any orthogonal matrix $V$.

The second stage of LDA/QR refines the first stage by addressing the issue of minimizing the within-class distance. It incorporates the within-class scatter information by applying a relaxation scheme on $V$ (relaxing $V$ from an orthogonal matrix to an arbitrary matrix). In the second stage of LDA/QR, we look for a transformation matrix $G$ such that $G = QV$, for some $V$. Note that $V$ is not required to be orthogonal. The original problem of computing $G$ is equivalent to computing $V$. Since $G^t S_b G = V^t(Q^t S_b Q)V$, $G^t S_w G = V^t(Q^t S_w Q)V$, and $G^t S_t G = V^t(Q^t S_t Q)V$, the original problem of finding optimal $G$ is equivalent to finding $V$, with $B = Q^t S_b Q$, $W = Q^t S_w Q$, and $T = Q^t S_t Q$ as the "reduced" between-class, within-class and total scatter matrices, respectively. Note that $B$ has much smaller size than the original scatter matrix $S_b$ (similarly for $W$ and $T$).

The optimal $V$ can be computed efficiently using many existing LDA-based methods, since we are dealing with matrices $B$, $W$, and $T$ of size $c$ by $c$. We can compute the optimal $V$ by simply applying regularized LDA; that is, we compute $V$, by solving a small eigenvalue problem on $(W + \mu I_c)^{-1} B$ or $(T + \mu I_c)^{-1} B$ (note $T = B + W$), for some positive constant $\mu$ [3]. The pseudo-code for this algorithm is given in **Algorithm 1**. We use the total scatter instead of the within-class scatter in Lines 4, 6, and 7, mainly for convenience of presentation of the kernel methods in Section 3 and Section 4.

## 3  Kernel discriminant analysis via QR-decomposition (KDA/QR)

In this section, the KDA/QR algorithm, a nonlinear extension of LDA/QR through kernel functions, is presented. Let $\Phi$ be a mapping to the feature space and $\Phi(A)$ be the data matrix in the feature space. Then, the centroid matrix $C^\Phi$ in the feature space is

$$C^\Phi = [m_1^\Phi, \cdots, m_c^\Phi] = \left[ \frac{1}{n_1} \sum_{i \in \Pi_1} \Phi(a_i), \cdots, \frac{1}{n_c} \sum_{i \in \Pi_c} \Phi(a_i) \right]. \tag{2}$$

The global centroid in the feature space can be computed as $m^\Phi = \frac{1}{n} \sum_i n_i m_i^\Phi$. To maximize between-class distance in the feature space, as discussed in Section 2, we perform QR decomposition on $C^\Phi$, i.e., $C^\Phi = Q^\Phi R^\Phi$. A key observation is that $R^\Phi$ can be computed as $(C^\Phi)^t C^\Phi = (R^\Phi)^t R^\Phi$ by applying the Cholesky decomposition on $(C^\Phi)^t C^\Phi$ [4].

Note that $C^\Phi = A^\Phi M$, where $A^\Phi = \Phi(A) = [\Phi(a_1) \ldots \Phi(a_n)]$, and the $i$th column of $M$ is $(0, \cdots, 0, \frac{1}{n_i}, \cdots, \frac{1}{n_i}, 0, \cdots, 0)^t$. Let $K$ be the kernel matrix with $K(i,j) = \langle \Phi(a_i), \Phi(a_j) \rangle$. Then

$$(C^\Phi)^t C^\Phi = M^t K M. \tag{3}$$

---

**Algorithm 2: KDA/QR**
/* Stage I: */
1.  Construct kernel matrix $K$;
2.  Compute $(C^\Phi)^t C^\Phi = M^t(KM)$ as in Eq. (3);
3.  Compute $R^\Phi$ from the Cholesky Decomposition of $(C^\Phi)^t C^\Phi$;
/* Stage II: */
4.  $Y^\Phi \leftarrow N^t M^t K M (R^\Phi)^{-1}$;
5.  $Z^\Phi \leftarrow E^t K M (R^\Phi)^{-1}$;
6.  $B^\Phi \leftarrow (Y^\Phi)^t Y^\Phi$;
7.  $T^\Phi \leftarrow (Z^\Phi)^t Z^\Phi$;
8.  Compute the $c$ eigenvectors $\phi_i^\Phi$ of $(T^\Phi + \mu I_c)^{-1} B^\Phi$, with decreasing eigenvalues;
9.  $V^\Phi \leftarrow [\phi_1^\Phi, \phi_2^\Phi, \cdots, \phi_c^\Phi]$;
10. $G^\Phi \leftarrow C^\Phi (R^\Phi)^{-1} V^\Phi$;

---

With the computed $R^\Phi$, $Q^\Phi = C^\Phi (R^\Phi)^{-1}$. The matrices $Y^\Phi$, $Z^\Phi$, $B^\Phi$, and $W^\Phi$ in the feature space (corresponding to the second stage in LDA/QR) can be computed as follows.

In the feature space, we have $H_b^\Phi = C^\Phi N$, where the $i^{th}$ column of $N$ is $((0, \cdots, \sqrt{n_i}, \cdots 0)^t - \frac{\sqrt{n_i}}{n}(n_1, \cdots, n_c)^t$. It follows that $Y^\Phi = (H_b^\Phi)^t Q^\Phi = N^t (C^\Phi)^t C^\Phi (R^\Phi)^{-1} = N^t M^t K M (R^\Phi)^{-1}$. Similarly, $H_t^\Phi = A^\Phi E$ and $Z^\Phi = (H_t^\Phi)^t Q^\Phi = E^t (A^\Phi)^t C^\Phi (R^\Phi)^{-1} = E^t (A^\Phi)^t A^\Phi M (R^\Phi)^{-1} = E^t K M (R^\Phi)^{-1}$, where $E = I - \frac{1}{n} e e^t$.

Since $S_b^\Phi = H_b^\Phi (H_b^\Phi)^t$ and $S_t^\Phi = H_t^\Phi (H_t^\Phi)^t$, we have

$$B^\Phi = (Q^\Phi)^t S_b^\Phi Q^\Phi = (Q^\Phi)^t H_b^\Phi (H_b^\Phi)^t Q^\Phi = (Y^\Phi)^t Y^\Phi,$$
$$T^\Phi = (Q^\Phi)^t S_t^\Phi Q^\Phi = (Q^\Phi)^t H_t^\Phi (H_t^\Phi)^t Q^\Phi = (Z^\Phi)^t Z^\Phi.$$

We proceed by computing the $c$ eigenvectors $\{\phi_i^\Phi\}_{i=1}^c$ of $(T^\Phi + \mu I_c)^{-1} B^\Phi$. Define $V^\Phi = [\phi_1^\Phi, \phi_2^\Phi, \cdots, \phi_c^\Phi]$. The final transformation matrix can be computed as

$$G^\Phi = Q^\Phi V^\Phi = C^\Phi (R^\Phi)^{-1} V^\Phi. \tag{4}$$

For a given data point $z$, its projection by $G^\Phi$ is $(G^\Phi)^t \Phi(z) = (V^\Phi)^t((R^\Phi)^{-1})^t (C^\Phi)^t \Phi(z) = (V^\Phi)^t((R^\Phi)^{-1})^t M^t K_{tz}$, where $K_{tz} \in \mathbb{R}^n$ and $K_{tz}(i) = \langle \Phi(a_i), \Phi(z) \rangle$.

The pseudo-code for the KDA/QR algorithm is given in **Algorithm 2**.

### 3.1 Complexity analysis of KDA/QR

The cost to formulate the kernel matrix in Line 1 is $O(n^2 d)$. The computation of $(C^\Phi)^t C^\Phi$ in Line 2 takes $O(n^2)$, taking advantage of the sparse structure of $M$. The Cholesky decomposition in Line 3 takes $O(c^3)$ [4]. Lines 4 takes $O(c^3)$, as $M^t K M$ is already computed in Line 2. In Line 5, the computation of $Z^\Phi = E^t K M (R^\Phi)^{-1} = (I - \frac{1}{n} e e^t) K M (R^\Phi)^{-1} = K M (R^\Phi)^{-1} - \frac{1}{n}\left(e\left((e^t K M)(R^\Phi)^{-1}\right)\right)$ in the given order takes $O(nc^2)$, assuming $KM$ is kept in Line 2. Lines 6, 7, and 8 take $O(c^3)$, $O(nc^2)$ and $O(c^3)$, respectively. Hence, the total complexity of the kernel LDA/QR algorithm is $O(n^2 d)$. Omitting the cost for evaluating the kernel matrix $K$, which is required in all kernel-based algorithms, the total cost is $O(n^2)$. Note that all other general discriminant analysis algorithms scale as $O(n^3)$.

## 4 Approximate KDA/QR (AKDA/QR)

In this section, we present the AKDA/QR algorithm, which is an efficient approximation of the KDA/QR algorithm from the last section. Note that the bottleneck of KDA/QR is the explicit formation of the large kernel matrix $K$ for the computation of $(C^\Phi)^t C^\Phi$ in Line 2 of **Algorithm 2**. The AKDA/QR algorithm presented in this section avoids the explicit construction of $K$, thus reducing the computational cost significantly.

The key to AKDA/QR is the efficient computation of $(C^\Phi)^t C^\Phi$, where $C^\Phi = [m_1^\Phi, \cdots, m_c^\Phi]$ and $m_j^\Phi = \frac{1}{n_j} \sum_{i \in \Pi_j} \Phi(a_i)$. AKDA/QR aims to find $x_j^*$ in the original space such that $\Phi(x_j^*)$ approximates $m_j^\Phi$. Mathematically, the optimal $x_j^*$ can be computed by solving the following optimization problem:

$$\min_{x_j \in R^d} \|\Phi(x_j) - \frac{1}{n_j} \sum_{i \in \Pi_j} \Phi(a_i)\|^2 \quad \text{for } j = 1, \cdots, c. \tag{5}$$

To proceed, we only consider Gaussian kernels for AKDA/QR, as they are the most widely used ones in the literature [9]. Furthermore, the optimization problem in (5) can be simplified by focusing on the Gaussian kernels, as shown in the following lemma.

**Lemma 4.1.** *Consider Gaussian kernel function $exp(-\|x - y\|^2/\sigma)$, where $\sigma$ is the bandwidth parameter. The optimization problem in (5) is convex if*

$$\text{for each } j = 1, \cdots, c \quad \text{and for all } i \in \Pi_j, \quad \|\sqrt{\frac{2}{\sigma}}(x_j - a_i)\| \leq 1 \tag{6}$$

*Proof.* It is easy to check that, for the Gaussian kernel, the optimization problem in (5) reduces to:

$$\min_{x_j \in R^d} f(x_j) \quad \text{for } j = 1, \cdots, c, \tag{7}$$

where $f(x) = \sum_{i \in \Pi_j} f_i(x)$ and $f_i(x) = -exp(-\|x - a_i\|^2/\sigma)$. The Hessian matrix of $f_i(x)$ is $H(f_i) = \frac{2}{\sigma} exp(-\|x - a_i\|^2/\sigma)(I - \frac{2}{\sigma}(x - a_i)(x - a_i)^t)$. It is easy to show that if $\|\sqrt{\frac{2}{\sigma}}(x - a_i)\| \leq 1$, for all $i \in \Pi_j$, then $H(f_i)$ is positive semi-definite, that is, $f_i(x)$ is convex. Thus, $f(x)$, the sum of convex functions is also convex. □

For applications involving high-dimensional data, such as face recognition, $\sigma$ is usually large (typically ranging from thousands to hundreds of thousands [13]), and the condition in Lemma 4.1 holds if we restrict our search space to the convex hull of each class in the original space. Therefore, the global minimum of the optimization problem in (7) can be found very efficiently using Newton's or gradient decent methods. A key observation is that for relatively large $\sigma$, the centroid of each class in the original space will map very close to the centroid in the feature space [9], which can serve as the approximate solution of the optimization problem in (7). Experiments show that choosing $x_j^* = \frac{1}{n_j} \sum_{i \in \Pi_j} a_i$ produces results close to the one by solving the optimization problem in (7). We thus use it in all the following experiments.

With the computed $x_j^*$, for $j = 1, \ldots, c$, the centroid matrix $C^\Phi$ can be approximated by

$$C^\Phi \approx [\Phi(x_1^*) \ldots \Phi(x_c^*)] (\equiv \hat{C}^\Phi) \tag{8}$$

and

$$(\hat{C}^\Phi)^t \hat{C}^\Phi = \hat{K}, \tag{9}$$

**Algorithm 3: AKDA/QR**

/* Stage I: */

1. Compute $x_j^* = \frac{1}{n_j} \sum_{i \in \Pi_j} a_i$, for $j = 1, \cdots, c$;
2. Construct kernel matrix $\hat{K}$ as in Eq. (9);
3. Compute $\hat{R}^\Phi$ from the Cholesky Decomposition of $\hat{K}$;

/* Stage II: */

4. $\hat{Y}^\Phi \leftarrow N^t \hat{K} (\hat{R}^\Phi)^{-1}$;
5. $\hat{Z}^\Phi \leftarrow E^t \hat{K}_{tc} (\hat{R}^\Phi)^{-1}$;
6. $\hat{B}^\Phi \leftarrow (\hat{Y}^\Phi)^t \hat{Y}^\Phi$;
7. $\hat{T}^\Phi \leftarrow (\hat{Z}^\Phi)^t \hat{Z}^\Phi$;
8. Compute the $c$ eigenvectors $\hat{\phi}_i^\Phi$ of $(\hat{T}^\Phi + \mu I_c)^{-1} \hat{B}^\Phi$, with decreasing eigenvalues;
9. $\hat{V}^\Phi \leftarrow [\hat{\phi}_1^\Phi, \hat{\phi}_2^\Phi, \cdots, \hat{\phi}_c^\Phi]$;
10. $\hat{G}^\Phi \leftarrow \hat{C}^\Phi (\hat{R}^\Phi)^{-1} \hat{V}^\Phi$;

|  | PCA | LDA/QR | GDA | KDA/QR | AKDA/QR |
|---|---|---|---|---|---|
| time | $O(n^2 d)$ | $O(ndc)$ | $O(n^2 d + n^3)$ | $O(n^2 d)$ | $O(ndc)$ |
| space | $O(nd)$ | $O(nc)$ | $O(n^2)$ | $O(n^2)$ | $O(nc)$ |

Table 1: Comparison of time & space complexities of several dimension reduction algorithms: $n$ is the size of the data, $d$ is the dimension, and $c$ is the number of classes.

where $\hat{K}(i,j) = \langle \Phi(x_i^*), \Phi(x_j^*) \rangle$ and $\hat{K} \in R^{c \times c}$. The Cholesky decomposition of $\hat{K}$ will give us $\hat{R}^\Phi$ by $\hat{K} = (\hat{R}^\Phi)^t \hat{R}^\Phi$.

It follows that $\hat{H}_b^\Phi = \hat{C}^\Phi N$, and $\hat{Y}^\Phi = N^t \hat{K} (\hat{R}^\Phi)^{-1}$. Similarly, $\hat{Z}^\Phi = E^t \hat{K}_{tc} (\hat{R}^\Phi)^{-1}$, where $N$ and $E$ are defined as in Section 3, and $\hat{K}_{tc}(i,j) = \langle \Phi(a_i), \Phi(x_j^*) \rangle$.

The following steps will be the same as the KDA/QR algorithm. The pseudo-code for AKDA/QR is given in **Algorithm 3**.

### 4.1 Complexity analysis of AKDA/QR

It takes $O(dn)$ in Line 1. The construction of the matrix $\hat{K}$ in Line 2 takes $O(c^2 d)$. The Cholesky Decomposition in Line 3 takes $O(c^3)$ [4]. Lines 4 and 5 take $O(c^3)$ and $O(ndc)$ respectively. It then takes $O(c^3)$ and $O(nc^2)$ for matrix multiplications in Lines 6 and 7, respectively. Line 8 computes the eigen-decomposition of a $c$ by $c$ matrix, hence takes $O(c^3)$ [4]. Thus, the most expensive step in **Algorithm 3** is Line 5, which takes $O(ndc)$.

Table 1 lists the time and space complexities of several dimension reduction algorithms. It is clear from the table that AKDA/QR is more efficient than other kernel based methods.

## 5 Experimental results

In this section, we evaluate both the KDA/QR and AKDA/QR algorithms. The performance is measured by classification accuracy. Note that both KDA/QR and AKDA/QR have two parameters: $\sigma$ for the kernel function and $\mu$ for the regularization. Experiments show that choosing $\sigma = 100000$ and $\mu = 0.15$ for KDA/QR, and $\sigma = 100000$ and $\mu = 0.10$ for AKDA/QR produce good overall results. We thus use these values in all the experiments. 1-Nearest Neighbor (1-NN) method is used as the classifier. We randomly select $p$ samples of each person from the dataset for training and the rest for

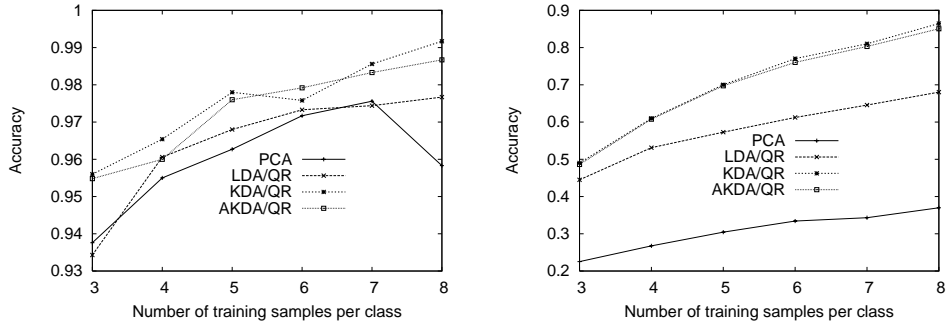

Figure 1: Comparison of classification accuracy on PIX (left) and AR (right).

testing. We repeat the experiments 20 times and report the average recognition accuracy of each method. The MATLAB codes for the KDA/QR and AKDA/QR algorithms may be accessed at http://www.cs.umn.edu/~jieping/Kernel.

**Datasets:** We use the following three datasets in our study, which are publicly available: PIX contains 300 face images of 30 persons. The image size of PIX image is $512 \times 512$. We subsample the images down to a size of $100 \times 100 = 10000$; ORL is a well-known dataset for face recognition. It contains ten different face images of 40 persons, for a total of 400 images. The image size is $92 \times 112 = 10304$; AR is a large face image datasets. We use a subset of AR. This subset contains 1638 face images of 126 persons. Its image size is $768 \times 576$. We subsample the images down to a size of $60 \times 40 = 2400$. Each dataset is normalized to have zero mean and unit variance.

**KDA/QR and AKQA/QR vs. LDA/QR:** In this experiment, we compare the performance of AKDA/QR and KDA/QR with that of several other linear dimension reduction algorithms including PCA, LDA/QR on two face datasets. We use 100 principal components for PCA as it produces good overall results. The results are summarized in Fig. 1, where the $x$-axis denotes the number of samples per class in the training set and the $y$-axis denotes the classification accuracy. Fig. 1 shows that KDA/QR and AKQA/QR consistently outperform LDA/QR and PCA. The most interesting result lies in the AR dataset, where AKDA/QR and KDA/QR outperform LDA/QR by a large margin. It is known that the images in the AR dataset contain pretty large area of occlusion due to sun glasses and scarves, which makes linear algorithms such as LDA/QR less effective. Another interesting observation is that the approximate AKQA/QR algorithm is competitive with its exact version KDA/QR in all cases.

**KDA/QR and AKQA/QR vs. GDA:** In this experiment, we compare the performance of AKDA/QR and KDA/QR with Generalized Discriminant Analysis (GDA) [1]. The comparison is made on the ORL face dataset, as the result of GDA on ORL is available in [5]. We also include the results on PCA and LDA/QR. The results are summarized in Table 2. The main observation from this experiment is that both KDA/QR and AKDA/QR are competitive with GDA, while AKDA/QR is much more efficient than GDA (see Table 1). Similar to the first experiment, Table 2 shows that KDA/QR and AKDA/QR consistently outperform the PCA and LDA/QR algorithms in terms of recognition accuracy.

## 6 Conclusions

In this paper, we first present a general kernel discriminant analysis algorithm, called KDA/QR. Using Gaussian kernels, we then proposed an approximate algorithm to

| p | PCA | LDA/QR | GDA | KDA/QR | AKDA/QR |
|---|------|--------|--------|--------|---------|
| 3 | 0.8611 | 0.8561 | 0.8782 | 0.9132 | 0.9118 |
| 4 | 0.8938 | 0.9083 | 0.9270 | 0.9321 | 0.9300 |
| 5 | 0.9320 | 0.9385 | 0.9535 | 0.9625 | 0.9615 |
| 6 | 0.9512 | 0.9444 | 0.9668 | 0.9737 | 0.9744 |
| 7 | 0.9633 | 0.9692 | 0.9750 | 0.9825 | 0.9815 |
| 8 | 0.9713 | 0.9713 | 0.9938 | 0.9875 | 0.9875 |

Table 2: Comparison of classification accuracy on ORL face image dataset. $p$ is the number of training samples per class. The results on GDA are taken from [5].

KDA/QR, which we call AKDA/QR. Our experimental results show that the accuracy achieved by the two algorithms is very competitive with GDA, a general kernel discriminant algorithms, while AKDA/QR is much more efficient. In particular, the computational complexity of AKDA/QR is linear in the number of the data points in the training set as well as the number of dimensions and the number of classes.

**Acknowledgment** Research of J. Ye and R. Janardan is sponsored, in part, by the Army High Performance Computing Research Center under the auspices of the Department of the Army, Army Research Laboratory cooperative agreement number DAAD19-01-2-0014, the content of which does not necessarily reflect the position or the policy of the government, and no official endorsement should be inferred.

## Footnotes

[1]KDA/QR stands for Kernel Discriminant Analysis via QR-decomposition

# References

[1] G. Baudat and F. Anouar. Generalized discriminant analysis using a kernel approach. *Neural Computation*, 12(10):2385–2404, 2000.

[2] P.N. Belhumeour, J.P. Hespanha, and D.J. Kriegman. Eigenfaces vs. fisherfaces: Recognition using class specific linear projection. *IEEE TPAMI*, 19(7):711–720, 1997.

[3] K. Fukunaga. *Introduction to Statistical Pattern Classification*. Academic Press, San Diego, California, USA, 1990.

[4] G. H. Golub and C. F. Van Loan. *Matrix Computations*. The Johns Hopkins University Press, Baltimore, MD, USA, third edition, 1996.

[5] Q. Liu, R. Huang, H. Lu, and S. Ma. Kernel-based optimized feature vectors selection and discriminant analysis for face recognition. In *ICPR Proceedings*, pages 362 – 365, 2002.

[6] S. Mika, G. Rätsch, and K.-R. Müller. A mathematical programming approach to the kernel fisher algorithm. In *NIPS Proceedings*, pages 591 – 597, 2001.

[7] S. Mika, G. Ratsch, J. Weston, B. Schökopf, and K.-R. Müller. Fisher discriminant analysis with kernels. In *IEEE Neural Networks for Signal Processing Workshop*, pages 41 – 48, 1999.

[8] S. Mika, A.J. Smola, and B. Schölkopf. An improved training algorithm for kernel fisher discriminants. In *AISTATS Proceedings*, pages 98–104, 2001.

[9] B. Schökopf and A. Smola. *Learning with Kernels: Support Vector Machines, Regularization, Optimization and Beyond*. MIT Press, 2002.

[10] B. Schökopf, A. Smola, and K. Müller. Nonlinear component analysis as a kernel eigenvalue problem. *Neural Computation*, 10(5):1299–1319, 1998.

[11] J. Ye and Q. Li. LDA/QR: An efficient and effective dimension reduction algorithm and its theoretical foundation. *Pattern recognition*, pages 851–854, 2004.

[12] J. Ye, Q. Li, H. Xiong, H. Park, R. Janardan, and V. Kumar. IDR/QR: An incremental dimension reduction algorithm via QR decomposition. In *ACM SIGKDD Proceedings*, pages 364–373, 2004.

[13] W. Zheng, L. Zhao, and C. Zou. A modified algorithm for generalized discriminant analysis. *Neural Computation*, 16(6):1283–1297, 2004.